# Structured Determinantal Point Processes

**Alex Kulesza**     **Ben Taskar**
Department of Computer and Information Science
University of Pennsylvania
Philadelphia, PA 19104
{kulesza,taskar}@cis.upenn.edu

## Abstract

We present a novel probabilistic model for distributions over *sets of structures*— for example, sets of sequences, trees, or graphs. The critical characteristic of our model is a preference for *diversity*: sets containing dissimilar structures are more likely. Our model is a marriage of structured probabilistic models, like Markov random fields and context free grammars, with determinantal point processes, which arise in quantum physics as models of particles with repulsive interactions. We extend the determinantal point process model to handle an exponentially-sized set of particles (structures) via a natural factorization of the model into parts. We show how this factorization leads to tractable algorithms for exact inference, including computing marginals, computing conditional probabilities, and sampling. Our algorithms exploit a novel polynomially-sized dual representation of determinantal point processes, and use message passing over a special semiring to compute relevant quantities. We illustrate the advantages of the model on tracking and articulated pose estimation problems.

## 1   Introduction

The need for distributions over *sets of structures* arises frequently in computer vision, computational biology, and natural language processing. For example, in multiple target tracking, sets of structures of interest are multiple object trajectories [6]. In gene finding, sets of structures of interest are multiple proteins coded by a single gene via alternative splicing [13]. In machine translation, sets of structures of interest are multiple interpretations or parses of a sentence in a different language [12]. Consider as a running example the problem of detecting and tracking several objects of the same type (e.g., cars, people, faces) in a video, assuming the number of objects is not known *a priori*. We would like a distribution over sets of trajectories that (1) includes sets of different cardinality and (2) prefers sets of trajectories that are spread out in space-time, as objects are likely to be [11, 15].

Determinantal point processes [10] are attractive models for distributions over sets, because they concisely capture *probabilistic mutual exclusion* between items via a kernel matrix that determines which items are similar and therefore less likely to appear together. Intuitively, the model balances the *diversity* of a set against the *quality* of the items it contains (for example, observation likelihood of an object along the trajectory, or motion smoothness). Remarkably, algorithms for computing certain marginal and conditional probabilities as well as sampling from this model are $O(N^3)$, where $N$ is total number of possible items, even though there are $2^N$ possible subsets of a set of size $N$ [7, 1] .

The problem, however, is that in our setting the total number of possible trajectories $N$ is exponential in the number of time steps. More generally, we consider modeling distributions over sets of structures (e.g., sequences, trees, graphs) where the total number of possible structures is exponential. Our structured determinatal point process model (SDPP) captures such distributions by combining structured probabilistic models (e.g., a Markov random field to model individual trajectory quality)

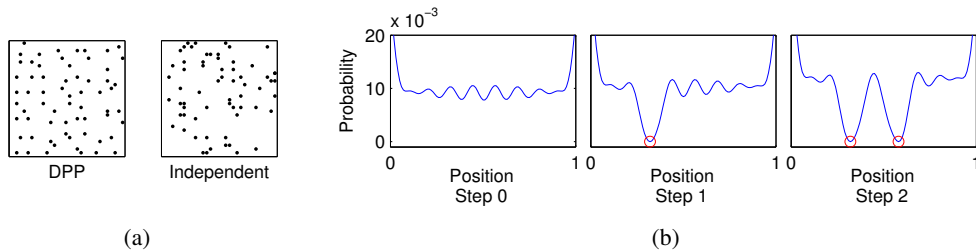

Figure 1: (a) A set of points in the plane drawn from a DPP (left), and the same number of points sampled independently (right). (b) The first three steps of sampling a DPP on a set of one-dimensional particle positions, from left to right. Red circles indicate already selected positions. The DPP naturally reduces the probabilities for positions that are similar to those already selected.

with determinantal point processes. We introduce a natural factorization of the determinantal model into parts (as in graphical models and grammars), and show that this factorization together with a novel dual representation of the process enables tractable inference and sampling using message passing algorithms over a special semiring. The contributions of this paper are: (1) introducing SDPPs, (2) a concise dual representation of determinantal processes, (3) tractable message passing algorithms for exact inference and sampling in SDPPs, (4) experimental validation on synthetic motion tracking and real-world pose detection problems. The paper is organized as follows: we present background on determinantal processes in Section 2 and introduce our model in Section 3; we develop inference and sampling algorithms in Section 4, and we describe experiments in Section 5.

## 2   Background: determinantal point processes

A point process $\mathcal{P}$ on a discrete set $\mathcal{Y} = \{y_1, \ldots, y_N\}$ is a probability measure on $2^{\mathcal{Y}}$, the set of all subsets of $\mathcal{Y}$. $\mathcal{P}$ is called a determinantal point process (DPP) if there exists a positive semidefinite matrix $K$ indexed by the elements of $\mathcal{Y}$ such that if $Y \sim \mathcal{P}$ then for every $A \subseteq \mathcal{Y}$, we have

$$\textbf{Determinantal Point Process:} \qquad \mathcal{P}(A \subseteq Y) = \det(K_A) \,. \qquad (1)$$

Here $K_A = [K_{ij}]_{y_i, y_j \in A}$ is the restriction of $K$ to the entries indexed by elements of $A$, and we adopt $\det(K_\emptyset) = 1$. We will refer to $K$ as the marginal kernel, as it contains all the information needed to compute the probability of including any subset $A$ in $Y \sim \mathcal{P}$. A few simple observations follow from Equation (1):

$$
\begin{aligned}
\mathcal{P}(y_i \in Y) &= K_{ii} & (2) \\
\mathcal{P}(y_i, y_j \in Y) &= K_{ii} K_{jj} - K_{ij} K_{ji} = \mathcal{P}(y_i \in Y)\mathcal{P}(y_j \in Y) - K_{ij}^2. & (3)
\end{aligned}
$$

That is, the diagonal of $K$ gives the marginal probabilities of inclusion for individual elements of $\mathcal{Y}$, and the off-diagonal elements determine the (anti-) correlations between pairs of elements: large values of $K_{ij}$ imply that $i$ and $j$ tend not to co-occur. Note that DPPs cannot represent distributions where elements are *more* likely to co-occur than if they were independent: correlations are negative.

Figure 1a shows the difference between sampling a set of points in the plane using a DPP (with $K_{ij}$ inversely related to the distance between points $i$ and $j$), which leads to a set that is spread out with good coverage, and sampling points independently, where the points exhibit random clumping.

Determinantal point processes, introduced to model fermions [10], also arise in studies of non-intersecting random paths, random spanning trees, and eigenvalues of random matrices [3, 2, 7]. The most relevant construction of DPPs for our purpose is via L-ensembles [1]. An **L-ensemble** defines a DPP via a positive semidefinite matrix $L$ indexed by the elements of $\mathcal{Y}$.

$$\textbf{L-ensemble DPP:} \qquad \mathcal{P}_L(Y) = \frac{\det(L_Y)}{\det(L + I)} \,, \qquad (4)$$

where $I$ is the $N \times N$ identity matrix. Note that $\mathcal{P}_L$ is normalized due to the identity $\sum_{Y \subseteq \mathcal{Y}} \det(L_Y) = \det(L + I)$. L-ensembles directly define the probability of observing each subset

of $\mathcal{Y}$, and subsets that have higher diversity (as measured by the corresponding determinant) have higher likelihood. To get probabilities of item co-occurrence as in Equation (1), we can compute the marginal kernel $K$ for the L-ensemble $\mathcal{P}_L$:

$$\text{\textbf{L-ensemble marginal kernel:}} \qquad K = (L + I)^{-1}L. \qquad (5)$$

Note that $K$ can be computed from the eigen-decomposition of $L = \sum_{k=1}^{N} \lambda_k v_k v_k^\top$ by a simple re-scaling of eigenvalues: $K = \sum_{k=1}^{N} \frac{\lambda_k}{\lambda_k+1} v_k v_k^\top$.

To get a better understanding of how $L$ affects marginals $K$, note that $L$ can be written as a Gram matrix with $L(y_i, y_j) = q(y_i)\phi(y_i)^\top \phi(y_j)q(y_j)$ for $q(y_i) \geq 0$ and some "feature mapping" $\phi(y)$ : $\mathcal{Y} \mapsto \mathbb{R}^D$, where $D \leq N$ and $||\phi(y_i)||_2 = 1$. We can think of $q(y_i)$ as the "quality score" for item $y_i$ and $\phi(y_i)^\top \phi(y_j)$ as normalized "similarity" between items $y_i$ and $y_j$.

$$\text{\textbf{L-ensemble (L=quality*similarity):}} \qquad \mathcal{P}_L(Y) \propto \det(\phi(Y)^\top \phi(Y)) \prod_{y_i \in Y} q^2(y_i), \qquad (6)$$

where $\phi(Y)$ is a $D \times |Y|$ matrix with columns $\phi(y_i)$, $y_i \in Y$. We will use this quality*similarity based representation extensively below. Roughly speaking, $\mathcal{P}_L(y_i \in Y)$ *increases* monotonically with quality $q(y_i)$ and $\mathcal{P}_L(y_i, y_j \in Y)$ *decreases* monotonically with similarity $\phi(y_i)^\top \phi(y_j)$.

We briefly mention a few other efficiently computable quantities of DPPs [1]:

$$\text{\textbf{L-ensemble conditionals:}} \quad \mathcal{P}_L(Y = A \cup B \mid A \subseteq Y) = \frac{\det(L_{A \cup B})}{\det(L + I_{\mathcal{Y} \setminus A})}, \qquad (7)$$

where $I_{\mathcal{Y} \setminus A}$ is the matrix with ones in the diagonal entries indexed by elements of $\mathcal{Y} \setminus A$ and zeros everywhere else. Conditional marginal probabilities $\mathcal{P}_L(B \subseteq Y \mid A \subseteq Y)$ as well as inclusion/exclusion probabilities $\mathcal{P}_L(A \subseteq Y \wedge B \cap Y = \emptyset)$ can also be computed efficiently using eigen-decompositions of $L$ and related matrices.

**Sampling**

Sampling from $\mathcal{P}_L$ is also efficient [7]. Let $L = \sum_{k=1}^{N} \lambda_k v_k v_k^\top$ be an orthonormal eigen-decomposition, and let $e_i$ be the $i$th standard basis $N$-vector (all zeros except for a 1 in the $i$th position). Then the following algorithm samples $Y \sim \mathcal{P}_L$:

---
Initialize: $Y = \emptyset, V = \emptyset$;
Add each eigenvector $\boldsymbol{v}_k$ to $V$ independently with prob. $\frac{\lambda_k}{\lambda_k+1}$;
**while** $|V| > 0$ **do**
&emsp;&emsp;Select a $y_i$ from $\mathcal{Y}$ with $\Pr(y_i) = \frac{1}{|V|} \sum_{v \in V} (v^\top e_i)^2$;
&emsp;&emsp;Update $Y = Y \cup y_i$;
&emsp;&emsp;Compute $V_\perp$, an orthonormal basis for the subspace of $V$ orthogonal to $e_i$, and let $V = V_\perp$;
**end**
Return $Y$;

---
**Algorithm 1:** Sampling algorithm for L-ensemble DPPs.

This yields a natural and efficient procedure for sampling from $\mathcal{P}$ given an eigen-decomposition of $L$. It also offers some additional insights. Because the dimension of $V$ is reduced by one on each iteration of the loop, and because the initial dimension of $V$ is simply the number of selected eigenvectors in step one, the size of $Y$ is distributed as the number of successes in $N$ Bernoulli trials where trial $k$ succeeds with probability $\frac{\lambda_k}{\lambda_k+1}$. In particular, $|Y|$ cannot be larger than $\text{rank}(L)$, and $\mathbb{E}[|Y|] = \sum_{k=1}^{N} \frac{\lambda_k}{\lambda_k+1}$.

To get a feel for the sampling algorithm, it is useful to visualize the distributions used to select $y_i$ at each time step, and to see how they are influenced by previously chosen items. Figure 1b shows this progression for a simple DPP where $\mathcal{Y}$ is the set of points in $[0, 1]$, quality scores are uniformly 1, and the feature mapping is such that $\phi(y_i)^\top \phi(y_j) \propto \exp(-(y_i - y_j)^2)$—that is, points are more similar the closer together they are. Initially, the eigenvectors $V$ give rise to a fairly uniform distribution over points in $\mathcal{Y}$, but as each successive point is selected and $V$ is updated, the distribution shifts to avoid points near those already chosen.

| Symbol | Meaning |
|--------|---------|
| $\mathcal{Y}, Y, y_i, N$ | $\mathcal{Y}$ is the base set, $Y$ is a subset of $\mathcal{Y}$, $y_i$ is an element of $\mathcal{Y}$, $N$ is the size of $|\mathcal{Y}|$ |
| $L, L_Y$ | $L$ is a p.s.d. matrix defining $\mathcal{P}(Y) \propto \det(L_Y)$, $L_Y$ is a submatrix indexed by $Y$ |
| $K, K_A$ | $K$ is a p.s.d. matrix defining marginals via $\mathcal{P}(A \subseteq Y) = \det(K_A)$ |
| $q(y_i), \phi(y_i)$ | quality*similarity decomposition; $L_{ij} = q(y_i)\phi(y_i)^\top \phi(y_j)q(y_j)$, $\phi(y_j) \in \mathbb{R}^D$ |
| $B, C$ | $C = BB^\top$ is the dual of $L = B^\top B$; the columns of $B$ are $B_i = q(y_i)\phi(y_i)$ |
| $\alpha, y_{i\alpha}, y_\alpha$ | $\alpha$ is a factor of a structure; $y_{i\alpha}, y_\alpha$ index the relevant part of the structure |

Table 1: Summary of notation.

## 3 Structured determinantal point processes

DPPs are amazingly tractable distributions when $N$, the size of the base set $\mathcal{Y}$, is small. However, we are interested in defining DPPs over exponentially sized $\mathcal{Y}$. For example, consider the case where each $y_i$ is itself a sequence of length $T$: $y_i = (y_{i1}, \ldots, y_{iT})$, where $y_{it}$ is the state at time $t$ (e.g., the location of an object in the $t$-th frame of a video). Assuming there are $n$ states at each time $t$ and all state transitions are possible, there are $n^T$ possible sequences, so $N = n^T$.

In order to define a DPP over structures such as sequences or trees, we assume a factorization of the quality score $q(y_i)$ and similarity score $\phi(y_i)^\top \phi(y_j)$ into parts, similar to a graphical model decomposition. For a sequence, the scores can be naturally decomposed into factors that depend on the state $y_{it}$ at each time $t$ and the states $(y_{it}, y_{it+1})$ for each transition $(t, t+1)$. More generally, we assume a set of factors and use the notation $y_{i\alpha}$ to refer to the $\alpha$ part of the structure $y_i$ (similarly, we use $y_\alpha$ to refer to the $\alpha$ part of the structure $y$). We assume that quality decomposes multiplicatively and similarity decomposes additively, as follows. (As before, $L(y_i, y_j) = q(y_i)\phi(y_i)^\top \phi(y_j)q(y_j)$.)

**Structured DPP Factorization:**  $\quad q(y_i) = \prod_\alpha q(y_{i\alpha}) \quad \text{and} \quad \phi(y_i) = \sum_\alpha \phi(y_{i\alpha}).$  (8)

We argue that these are quite natural factorizations. Quality scores, for example, can be given by a typical log-linear Markov random field, which defines a multiplicative distribution over structures. Similarity scores can be thought of as dot products between features of the two labelings.

In our tracking example, the feature mapping $\phi(y_{it})$ should reflect similarity between trajectories; e.g., features could track coarse-level position at time $t$, so that the model considers sets with trajectories that pass near or through the same states less likely. A common problem in multiple target tracking is that the quality of one object's trajectory and its neighborhood "tube" is often much more likely than other objects' trajectories as measured by an HMM or CRF model, so standard sampling from a graphical model will produce very similar, overlapping trajectories, ignoring less "detectable" targets. A sample from the structured DPP model would be much more likely to contain diverse trajectories. (See Figure 2.)

**Dual representation**

While the factorization in Equation (8) concisely defines a DPP over a structured $\mathcal{Y}$, the more remarkable fact is that it gives rise to tractable algorithms for computing key marginals and conditionals when the set of factors is low-treewidth, just as in graphical model inference [8], even though $L$ is too large to even write down. We propose the following dual representation of $L$ in order to exploit the factorization. Let us define a $D \times N$ matrix $B$ whose columns are given by $B_i = q(y_i)\phi(y_i)$, so that $L = B^\top B$. Consider the $D \times D$ matrix $C = BB^\top$; note that typically $D \ll N$ (actually, the rank of $B$ is at most $O(nT)$ in the sequence case). The eigenvalues of $C$ and $L$ are identical, and the eigenvectors are related as follows: if $C = \sum_k \lambda_k v_k v_k^\top$, then $L = \sum_k \lambda_k (B^\top v_k)^\top (B^\top v_k)$. That is, if $v_k$ is the $k$-th eigenvector of $C$, $B^\top v_k$ is the $k$-th eigenvector of $L$, and it has the same eigenvalue $\lambda_k$. This connection allows us to compute important quantities from $C$.

For example, to compute the L-ensemble normalization $\det(L + I) = \prod_k (\lambda_k + 1)$ in Equation (4), we just need the eigenvalues of $C$. To compute $C$ itself, we need to compute $BB^\top = \sum_{y_i} q^2(y_i)\phi(y_i)\phi(y_i)^\top$. This appears daunting, but the factorization turns out to offer an efficient dynamic programming solution. We discuss in more detail how to compute $C$ for sequences (and for fixed-treewidth factors in general) in the next section. Assuming we can compute $C$ efficiently,

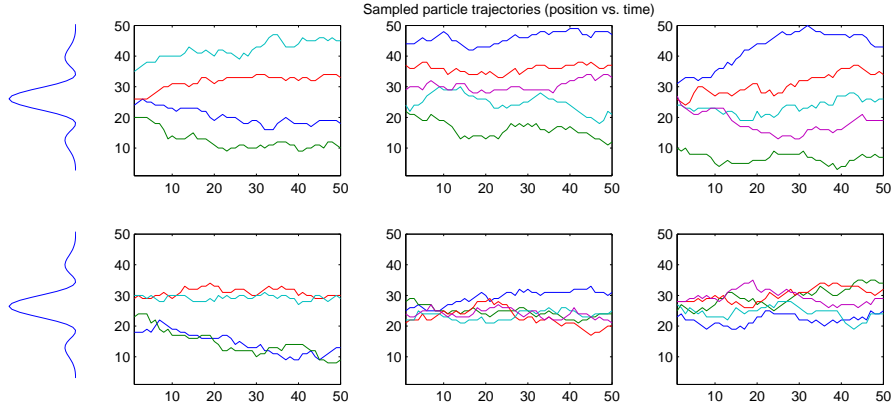

Figure 2: Sets of (structured) particle trajectories sampled from the SDPP (top row) and independently using only quality scores (bottom row). The curves to the left indicate the quality scores for the possible initial positions.

we can eigen-decompose it as $C = \sum_k \lambda_k v_k v_k^\top$ in $O(D^3)$. Then, to compute $\mathcal{P}_L(y_i \in Y)$, the probability of any single trajectory being included in $Y \sim \mathcal{P}_L$, we have all we need:

**Structured Marginal:**   $$K_{ii} = \sum_k \frac{\lambda_k}{\lambda_k + 1}(B_i^\top v_k)^2 = q^2(y_i) \sum_k \frac{\lambda_k}{\lambda_k + 1}(\phi(y_i)^\top v_k)^2 \qquad (9)$$

Similarly, given two trajectories $y_i$ and $y_j$, $\mathcal{P}_L(y_i, y_j \in Y) = K_{ii}K_{jj} - K_{ij}^2$, where:

$$K_{ij} = \sum_k \frac{\lambda_k}{\lambda_k + 1}(B_i^\top v_k)(B_j^\top v_k) = q(y_i)q(y_j) \sum_k \frac{\lambda_k}{\lambda_k + 1}(\phi(y_i)^\top v_k)(\phi(y_j)^\top v_k). \qquad (10)$$

## 4   Inference for SDPPs

We now turn to computing $C$ using the factorization in Equation (8). We have

$$C = \sum_{y \in \mathcal{Y}} q^2(y)\phi(y)\phi(y)^\top = \sum_{y \in \mathcal{Y}} \left( \prod_\alpha q^2(y_\alpha) \right) \left( \sum_\alpha \phi(y_\alpha) \right) \left( \sum_\alpha \phi(y_\alpha) \right)^\top. \qquad (11)$$

If we think of $q^2(y_\alpha)$ as factor potentials of a graphical model $p(y) \propto \prod_\alpha q^2(y_\alpha)$, then computing $C$ is equivalent to computing second moments of additive features (modulo normalization $Z$). A naive algorithm can simply compute all $O(T^2)$ pairwise marginals $p(y_\alpha, y_{\alpha'})$ and, by linearity of expectation, add up the contributions: $C = Z \sum_{\alpha,\alpha'} \sum_{y_\alpha, y_{\alpha'}} p(y_\alpha, y_{\alpha'})\phi(y_\alpha)\phi(y_{\alpha'})^\top$.

However, we can use a much more efficient $O(D^2 T)$ algorithm based on second-order semiring message passing [9]. The details are given in Appendix A of the supplementary material, but in short we apply the standard two-pass belief propagation algorithm for trees with a particular semiring in place of the usual sum-product or max-sum. By performing message passing under this second-order semiring, one can efficiently compute any quantity of the form:

$$\sum_{y \in \mathcal{Y}} \left( \prod_\alpha p(y_\alpha) \right) \left( \sum_\alpha a(y_\alpha) \right) \left( \sum_\alpha b(y_\alpha) \right) \qquad (12)$$

for functions $p \geq 0$, $a$, and $b$ in time $O(T)$. Since the outer product in Equation (11) comprises $D^2$ quantities of the type in Equation (12), we can compute $C$ in time $O(D^2 T)$.

### Sampling

As described in Section 3, the eigen-decomposition of $C$ yields an implicit representation of $L$: for each eigenvalue/vector pair $(\lambda_k, v_k)$ of $C$, $(\lambda_k, B^\top v_k)$ is a corresponding pair for $L$. We show that this implicit representation is enough to efficiently perform the sampling procedure in Algorithm 1.

The key is to represent $V$, the orthonormal set of vectors in $\mathbb{R}^N$, as a set $\hat{V}$ of vectors in $\mathbb{R}^D$, with the mapping $V = \{B^\top v | v \in \hat{V}\}$. Let $v_i, v_j$ be two arbitrary vectors in $\hat{V}$. Then we have $(B^\top v_i)^\top (B^\top v_j) = v_i^\top BB^\top v_j = v_i^\top Cv_j$. Thus we can compute dot products between vectors in $V$ using their preimage in $\hat{V}$. This is sufficient to compute the normalization for each eigenvector $B^\top v$, as required to obtain an initial orthonormal basis. Trivially, we can also compute (implicit) sums between vectors in $V$; this combined with dot products is enough to perform the Gram-Schmidt orthonormalization needed to obtain $\hat{V}_\perp$ from $\hat{V}$ and the most recently selected $y_i$ at each iteration.

All that remains, then, is to choose a structure $y_i$ according to the distribution $\Pr(y_i) = 1/|\hat{V}| \sum_{v \in \hat{V}} ((B^\top v)^\top e_i)^2$. Recall that the columns of $B$ are given by $B_i = q(y_i)\phi(y_i)$. Thus the distribution can be rewritten as

$$\Pr(y_i) = \frac{1}{|\hat{V}|} \sum_{v \in \hat{V}} q^2(y_i)(v^\top \phi(y_i))^2 \,. \tag{13}$$

By assumption $q^2(y_i)$ decomposes multiplicatively over parts of $y_i$, and $v^\top \phi(y_i)$ decomposes additively. Thus the distribution is a sum of $|\hat{V}|$ terms, each having the form of Equation (12). We can therefore apply message passing in the second-order semiring to compute marginals of this distribution—that is, for each part $y_\alpha$ we can compute

$$\sum_{y \sim y_\alpha} \frac{1}{|\hat{V}|} \sum_{v \in \hat{V}} q^2(y)(v^\top \phi(y))^2 \,, \tag{14}$$

where the sum is over all structures consistent with the value of $y_\alpha$. This only takes $O(T|\hat{V}|)$ time.

In fact, the message-passing computation of these marginals yields an efficient algorithm for sampling individual full structures $y_i$ as required by Algorithm 1; the key is to pass normal messages forward, but *conditional* messages backward. Suppose we have a sequence model; since the forward pass completes with correct marginals at the final node, we can correctly sample its value before any backwards messages are sent. Once the value of the final node is fixed, we pass a conditional message backwards; that is, we send zeros for all values other than the one just selected. This results in condtional marginals at the penultimate node. We can then conditionally sample its value, and repeat this process until all nodes have been assigned. Furthermore, by applying the second-order semiring we are able to sample from a distribution quite different from that of a traditional graphical model. The algorithm is described in more detail in Appendix B of the supplementary material.

## 5  Experiments

We begin with a synthetic motion tracking task, where the goal is to follow a collection of particles as they travel in a one-dimensional space over time. This is the structured analog of the setting shown in Figure 1b, where elements of $\mathcal{Y}$ are no longer single positions in $[0, 1]$, but are now sequences of such positions over many time periods. For our experiments, we modeled paths $y_i$ over $T = 50$ time steps, where at each time $t$ a particle can be in one of 50 discretized positions, $y_{it} \in \{1, \ldots, 50\}$. The total number of possible trajectories is thus $50^{50}$, and there are $2^{50^{50}}$ possible sets of trajectories.

While a real tracking problem would involve quality scores $q(y)$ that depend on some observations, e.g., measurements over time from a set of physical sensors, for simplicity we determine the quality of a trajectory using only its starting position and a measure of smoothness over time: $q(y) = q(y_1) \prod_{t=2}^{T} q(y_{t-1}, y_t)$. The initial quality scores $q(y_1)$ depicted on the left of Figure 2 are high in the middle with secondary modes on each side. The transition quality is given by $q(y_{t-1}, y_t) = f(y_{t-1} - y_t)$, where $f$ is the density function of the zero-mean Gaussian with unit variance. We scale the quality scores so that the expected number of selected trajectories is 5.

We want trajectories to be considered similar if they travel through similar positions, so we define a 50-dimensional feature vector $\phi(y) = \sum_{t=1}^{T} \phi(y_t)$ where $\phi_r(y_t) \propto f(i - y_t)$ for $r = 1, \ldots, 50$. Intuitively, feature $r$ is activated when the trajectory passes near position $r$, so trajectories passing through nearby positions will activate the same features and thus appear similar.

Figure 2 shows the results of applying our SDPP sampling algorithm to this setting. Sets of trajectories drawn independently according to quality score tend to cluster in the middle region (second

row). The SDPP samples, however, are more diverse, tending to cover more of the space while still respecting the quality scores—they are still smooth, and still tend to start near the middle position.

**Pose estimation**

To demonstrate that SDPPs effectively model characteristics of real-world data, we apply them to a multiple-person pose estimation task. Our dataset consists of 73 still frames taken from various TV shows, each approximately 720 by 540 pixels in size[1]. As much as possible, the selected frames contain three or more people at similar scale, all facing the camera and without serious occlusions. Sample images from the dataset are shown in Figure 4. The task is to identify the location and pose of each person in the image. For our purposes, each pose is a structure containing four parts (head, torso, right arm, and left arm), each of which takes a value consisting of a pixel location and an orientation (one of 24 discretized angles). There are approximately 75,000 possible such values for each part, so there are about $4^{75,000}$ possible poses. Each image was labeled by hand for evaluation.

We use a standard pictorial strucure model [4, 5], treating each pose as a two-level tree with the torso as the root and the head and arms as leaves. Our quality scores are derived from [14]; they factorize across the nodes (body parts) $P$ and edges (joints) $J$ as $q(y) = \gamma(\prod_{p \in P} q(y_p) \prod_{pp' \in J} q(y_p, y_{p'}))^\beta$. $\gamma$ is a scale parameter that controls the expected number of poses in each sample, and $\beta$ is a sharpness parameter that we found helpful in controlling the impact of the quality scores. (We set parameter values using a held-out training set; see below.) Each part receives a quality score $q(y_p)$ given by a customized part detector previously trained on similar images. The joint quality score $q(y_p, y_{p'})$ is given by a Gaussian "spring" that encourages, for example, the left arm to begin near the left shoulder. Full details of the quality terms are provided in [14].

Given our data, we want to discourage the model from selecting overlapping poses, so we design our similarity features spatially. We define an evenly spaced 8 by 4 grid of reference points $x_1, \ldots, x_{32}$, and use $\phi(y) = \sum_{p \in P} \phi(y_p)$, where $\phi_r(y_p) \propto f(\|y_p - x_r\|_2/\sigma)$. Recall that $f$ is the standard normal density function, and $\|y_p - x_r\|_2$ is the distance between the position of part $p$ (ignoring angle) and the reference point $x_r$. The parameter $\sigma$ controls the width of the kernel. Poses that occupy the same part of the image will be near the same reference points, and thus appear similar.

We compare our model against two baselines. The first is an independent model which draws poses independently according to the distribution obtained by normalizing the quality scores. The second is a simple non-maxima suppression model that iteratively selects successive poses using the normalized quality scores, but under the hard constraint that they do not overlap with any previously selected pose. (Poses overlap if they cover any of the same pixels when rendered.) In both cases, the number of poses is given by a draw from the SDPP model, ensuring no systematic bias.

We split our data randomly into a training set of 13 images and a test set of 60 images. Using the training set, we select values for $\gamma$, $\beta$, and $\sigma$ that optimize overall $F_1$ score at radius 100 (see below), as well as distinct optimal values of $\beta$ for the baselines. ($\gamma$ and $\sigma$ are irrelevant for the baselines.) We then use each model to sample 10 sets of poses for each test image, or 600 samples per model.

For each sample, we compute precision, recall, and $F_1$ score. For our purposes, precision is the fraction of predicted parts where both endpoints are within a particular radius of the endpoints of an expert-labeled part of the same type (head, left arm, etc.). Correspondingly, recall is the fraction of expert-labeled parts within a given radius of a predicted part of the same type. Since our SDPP model encourages diversity, we expect to see improvements in recall at the expense of precision. $F_1$ score is the harmonic mean of precision and recall. We compute all metrics separately for each sample, and then average the results across samples and images in the test set.

The results over several different radii are shown in Figure 3a. At tight tolerances the SDPP performs comparably to the independent samples (perhaps because the quality scores are only accurate at the mode, so diverse samples are not close enough to be valuable). As the radius increases, however, the SDPP obtains significantly better results, outperforming both baselines. Figure 3b shows the curves for the arms alone; the arms tend to be more difficult to locate accurately. Figure 3c shows the precision/recall obtained by each model. As expected, the SDPP model achieves its improved $F_1$ score by increasing recall at the cost of precision.

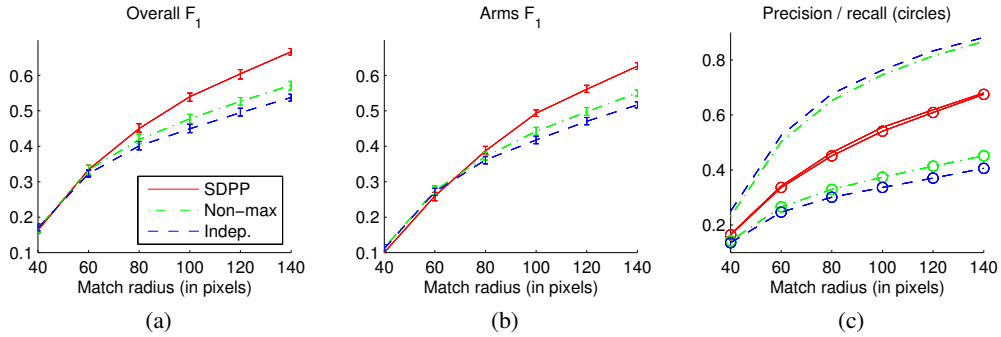

Figure 3: Results for pose estimation. The horizontal axis gives the distance threshold used to determine whether two parts are successfully matched. 95% confidence intervals are shown.

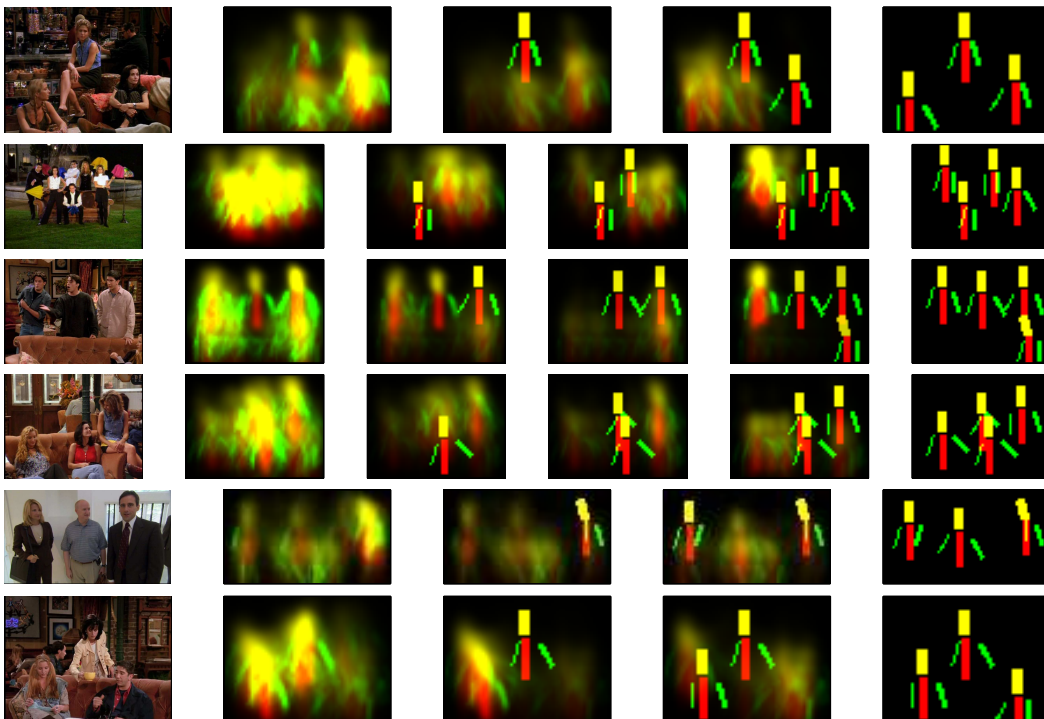

Figure 4: Structured marginals for the pose estimation task on successive steps of the sampling algorithm, with already selected poses superimposed. Input images are shown on the left.

For illustration, we show the sampling process for a few images in Figure 4. As in Figure 1b, the SDPP efficiently discounts poses that are similar to those already selected.

# 6 Conclusion

We introduced the structured determinantal point process (SDPP), a probabilistic model over sets of structures such as sequences, trees, or graphs. We showed the intuitive "diversification" properties of the SDPP, and developed efficient message-passing algorithms to perform inference through a dual characterization of the standard DPP and a natural factorization.

**Acknowledgments**

The authors were partially supported by NSF Grant 0803256.

## Footnotes

[1]The images and code from [14] are available at `http://www.vision.grasp.upenn.edu/video`

# References

[1] A. Borodin. Determinantal point processes, 2009.

[2] A. Borodin and A. Soshnikov. Janossy densities. I. Determinantal ensembles. *Journal of Statistical Physics*, 113(3):595–610, 2003.

[3] D. Daley and D. Vere-Jones. *An introduction to the theory of point processes: volume I: elementary theory and methods*. Springer, 2003.

[4] P. Felzenszwalb and D. Huttenlocher. Pictorial structures for object recognition. *International Journal of Computer Vision*, 61(1):55–79, 2005.

[5] M. Fischler and R. Elschlager. The representation and matching of pictorial structures. *IEEE Transactions on Computers*, 100(22), 1973.

[6] D. Forsyth and J. Ponce. *Computer Vision: A Modern Approach*. Prentice Hall, 2003.

[7] J. Hough, M. Krishnapur, Y. Peres, and B. Virág. Determinantal processes and independence. *Probability Surveys*, 3:206–229, 2006.

[8] D. Koller and N. Friedman. *Probabilistic Graphical Models: Principles and Techniques*. The MIT Press, 2009.

[9] Z. Li and J. Eisner. First-and second-order expectation semirings with applications to minimum-risk training on translation forests. In *Proc. EMNLP*, 2009.

[10] O. Macchi. The coincidence approach to stochastic point processes. *Advances in Applied Probability*, 7(1):83–122, 1975.

[11] J. MacCormick and A. Blake. A probabilistic exclusion principle for tracking multiple objects. *International Journal of Computer Vision*, 39(1):57–71, 2000.

[12] C. D. Manning and H. Schütze. *Foundations of Statistical Natural Language Processing*. MIT Press, Boston, MA, 1999.

[13] T. Nilsen and B. Graveley. Expansion of the eukaryotic proteome by alternative splicing. *Nature*, 463(7280):457–463, 2010.

[14] B. Sapp, C. Jordan, and B. Taskar. Adaptive pose priors for pictorial structures. In *IEEE Computer Society Conference on Computer Vision and Pattern Recognition (CVPR'10)*, 2010.

[15] T. Zhao and R. Nevatia. Tracking multiple humans in complex situations. *IEEE Transactions on Pattern Analysis and Machine Intelligence*, 26:1208–1221, 2004.

